# Constructing Distributed Representations Using Additive Clustering

**Wheeler Ruml**
Division of Engineering and Applied Sciences
Harvard University
33 Oxford Street, Cambridge, MA 02138
`ruml@eecs.harvard.edu`

## Abstract

If the promise of computational modeling is to be fully realized in higher-level cognitive domains such as language processing, principled methods must be developed to construct the semantic representations used in such models. In this paper, we propose the use of an established formalism from mathematical psychology, *additive clustering*, as a means of automatically constructing binary representations for objects using only pairwise similarity data. However, existing methods for the unsupervised learning of additive clustering models do not scale well to large problems. We present a new algorithm for additive clustering, based on a novel heuristic technique for combinatorial optimization. The algorithm is simpler than previous formulations and makes fewer independence assumptions. Extensive empirical tests on both human and synthetic data suggest that it is more effective than previous methods and that it also scales better to larger problems. By making additive clustering practical, we take a significant step toward scaling connectionist models beyond hand-coded examples.

## 1   Introduction

Many cognitive models posit mental representations based on discrete substructures. Even connectionist models whose processing involves manipulation of real-valued activations typically represent objects as patterns of 0s and 1s across a set of units (Noelle, Cottrell, and Wilms, 1997). Often, individual units are taken to represent specific features of the objects and two representations will share features to the degree to which the two objects are similar. While this arrangement is intuitively appealing, it can be difficult to construct the features to be used in such a model. Using random feature assignments clouds the relationship between the model and the objects it is intended to represent, diminishing the model's value. As Clouse and Cottrell (1996) point out, hand-crafted representations are tedious to construct and it can be difficult to precisely justify (or even articulate) the principles that guided their design. These difficulties effectively limit the number of objects that can be encoded, constraining modeling efforts to small examples. In this paper, we investigate methods for automatically synthesizing feature-based representations directly from the pairwise object similarities that the model is intended to respect. This automatic

Table 1: An 8-feature model derived from consonant confusability data. With $c = 0.024$, the model accounts for 91.8% of the variance in the data.

| Wt. | Objects with feature | Interpretation |
|---|---|---|
| .350 | fθ | front unvoiced fricatives |
| .243 | dg | back voiced stops |
| .197 | p k | unvoiced stops (without t) |
| .182 | b v∂ | front voiced |
| .162 | ptk | unvoiced stops |
| .127 | mn | nasals |
| .075 | dgv∂zž | voiced (without b) |
| .049 | ptkfθsš | unvoiced |

approach eliminates the manual burden of selecting and assigning features while providing an explicit design criterion that objectively connects the representations to empirical data.

After formalizing the problem, we will review existing algorithms that have been proposed for solving it. We will then investigate a new approach, based on combinatorial optimization. When using a novel heuristic search technique, we find that the new approach, despite its simplicity, performs better than previous algorithms and that, perhaps more important, it maintains its effectiveness on large problems.

## 1.1 Additive Clustering

We will formalize the problem of constructing discrete features from similarity information using the *additive clustering* model of Shepard and Arabie (1979). In this framework, abbreviated ADCLUS, clusters represent arbitrarily overlapping discrete features. Each of the $k$ features has a non-negative real-valued weight $w_k$, and the similarity between two objects $i$ and $j$ is just the sum of the weights of the features they share. If $f_{ik}$ is 1 if object $i$ has feature $k$ and 0 otherwise, and $c$ is a real-valued constant, then the similarity of $i$ and $j$ is modeled as

$$\hat{s}_{ij} = \sum_k w_k f_{ik} f_{jk} + c \ .$$

This class of models is very expressive, encompassing non-hierarchical as well as hierarchical arrangements of clusters. An example model, derived using the `ewindclus-klb` algorithm described below, is shown in Table 1. The representation of each object is simply the binary column specifying its membership or absence in each cluster. Additive clustering is asymmetric in the sense that only the shared features of two objects contribute to their similarity, not the ones they both lack. (This is the more general formulation, as an additional feature containing the set complement of the original feature could always be used to produce such an effect.)

With a model formalism in hand, we can then phrase the problem of constructing feature assignments as simply finding the ADCLUS model that best matches the given similarity data using the desired number of features. The fit of a model (comprising $F$, $W$, and $c$) to a matrix, $S$, can be quantified by the variance accounted for (VAF), which compares the model's accuracy to merely predicting using the mean similarity:

$$\text{VAF} = 1 - \frac{\sum_{i,j}(s_{ij} - \hat{s}_{ij})^2}{\sum_{i,j}(s_{ij} - \bar{s})^2}$$

A VAF of 0 can always be achieved by setting all $w_k$ to 0 and $c$ to $\bar{s}$.

## 2  Previous Algorithms

Additive clustering is a difficult 0-1 quadratic programming problem and only heuristic methods, which do not guarantee an optimal model, have been proposed. Many different approaches have been taken:

**Subsets:** Shepard and Arabie (1979) proposed an early algorithm based on subset analysis that was clearly superseded by Arabie's later work below. Hojo (1983) also proposed an algorithm along these lines. We will not consider these algorithms further.

**Non-discrete Approximation:** Arabie and Carroll (1980) and Carroll and Arabie (1983) proposed the two-stage `indclus` algorithm. In the first stage, cluster memberships are treated as real values and optimized for each cluster in turn by gradient descent. At the same time, a penalty term for non-0-1 values is gradually increased. Afterwards, a combinatorial clean-up stage tries all possible changes to 1 or 2 cluster memberships. Experiments reported below use the original code, modified slightly to handle large instances. Random initial configurations were used.

**Asymmetric Approximation:** In the `sindclus` algorithm, Chaturvedi and Carroll (1994) optimize an asymmetric model with two sets of cluster memberships, having the form $\hat{s}_{ij} = \sum_k w_k f_{ik} g_{jk} + c$. By considering each cluster in turn, this formulation allows a fast method for determining each of $F$, $G$, and $w$ given the other two. In practice, $F$ and $G$ often become identical, yielding an ADCLUS model. Experiments reported below use both a version of the original implementation that has been modified to handle large instances and a reimplemented version (`re-sindclus`) that differs in its behavior at boundary cases (handling 0 weights, empty clusters, ties). Models from runs in which $F$ and $G$ did not converge were each converted into several ADCLUS models by taking only $F$, only $G$, their intersection, or their union. The weights and constants of each model were optimized using constrained least-squares linear regression (Stark and Parker, 1995), ensuring non-negative cluster weights, and the one with the highest VAF was used.

**Alternating Clusters:** Kiers (1997) proposed an element-wise simplified `sindclus` algorithm, which we abbreviate as `ewindclus`. Like `sindclus`, it considers each cluster in turn, alternating between the weights and the cluster memberships, although only one set of clusters is maintained. Weights are set by a simple regression and memberships are determined by a gradient function that assumes object independence and fixed weights. The experiments reported below use a new implementation, similar to the reimplementation of `sindclus`.

**Expectation Maximization:** Tenenbaum (1996) reformulated ADCLUS fitting in probabilistic terms as a problem with multiple hidden factorial causes, and proposed a combination of the EM algorithm, Gibbs sampling, and simulated annealing to solve it. The experiments below use a modified version of the original implementation which we will notate as `em-indclus`. It terminates early if 10 iterations of EM pass without a change in the solution quality. (A comparison with the original code showed this modification to give equivalent results using less running time.)

Unfortunately, it is not clear which of these approaches is the best. Most published comparisons of additive clustering algorithms use only a small number of test problems (or only artificial data) and report only the best solution found within an unspecified amount of time. Because the algorithms use random starting configurations and often return solutions of widely varying quality even when run repeatedly on the same problem, this leaves it unclear which algorithm gives the best results on a typical run. Furthermore, different

Table 2: The performance of several previously proposed algorithms on data sets from psychological experiments.

| Name | indclus VAF | indclus IQR | sindclus VAF | sindclus IQR | sindclus $\overline{r}$ | re-sindclus VAF | re-sindclus IQR | re-sindclus $\overline{r}$ | ewindclus VAF | ewindclus IQR | ewindclus $\overline{r}$ |
|---|---|---|---|---|---|---|---|---|---|---|---|
| animals-s | 77 | 75–80 | 66 | 65–65 | 8 | 78 | 79–80 | 12 | 64 | 60–69 | 4 |
| numbers | 83 | 81–86 | 84 | 82–86 | 5 | 78 | 75–81 | 7 | 82 | 79–85 | 5 |
| workers | 83 | 82–85 | 81 | 79–83 | 9 | 84 | 82–85 | 7 | 67 | 63–72 | 2 |
| consonants | 89 | 89–90 | 88 | 87–89 | 6 | 81 | 80–82 | 5 | 51 | 44–57 | 1 |
| animals | 71 | 69–74 | 66 | 66–66 | 9 | 66 | 66–66 | 13 | 72 | 71–73 | 26 |
| letters | 80 | 80–80 | 78 | 78–79 | 7 | 68 | 65–72 | 5 | 74 | 73–75 | 17 |

Table 3: The performance of indclus and em-indclus on the human data sets.

| Name | $n$ | $k$ | indclus VAF | indclus IQR | indclus $\overline{r}$ | em-indclus VAF | em-indclus IQR |
|---|---|---|---|---|---|---|---|
| animals-s | 10 | 3 | 80 | 80–80 | 23 | 80 | 80–80 |
| numbers | 10 | 8 | 91 | 90–91 | 157 | 90 | 89–90 |
| workers | 14 | 7 | 89 | 88–89 | 89 | 87 | 87–89 |
| consonants | 16 | 8 | 91 | 91–91 | 291 | 91 | 91–91 |
| animals | 26 | 12 | 71 | 69–74 | 1 | N/A | |
| letters | 30 | 5 | 82 | 82–83 | 486 | 82 | 82–83 |

algorithms require very different running times, and multiple runs of a fast algorithm with high variance in solution quality may produce a better result in the same time as a single run of a more predictable algorithm. The next section reports on a new empirical comparison that addresses these concerns.

## 2.1 Evaluation of Previous Algorithms

We compared indclus, both implementations of sindclus, ewindclus, and em-indclus on 3 sets of problems. The first set is a collection of 6 typical data sets from psychological experiments that have been used in previous additive clustering work (originally by Shepard and Arabie (1979), except for animals-s, Mechelen and Storms (1995), and animals, Chaturvedi and Carroll (1994)). The number of objects ($n$) and the number of features used ($k$) are listed for each instance as part of Table 3. The second set of problems contains noiseless synthetic data derived from ADCLUS models with 8, 16, 32, 64, and 128 objects. In a rough approximation of the human data, the number of clusters was set to $2\log_2(n)$, and as in previous ADCLUS work, each object was inserted in each cluster with probability 0.5. A single similarity matrix was generated from each model using weights and constants uniformly distributed between 1 and 6. The third set of problems was derived from the second by adding gaussian noise with a variance of 10% of the variance of the similarity data and enforcing symmetry. Each algorithm was run at least 50 times on each data set. Runs that crashed or resulted in a VAF $< 0$ were ignored. To avoid biasing our conclusions in favor of methods requiring more computation time, those results were then used to derive the distribution of results that would be expected if all algorithms were run simultaneously and those that finished early were re-run repeatedly until the slowest algorithm finished its first run, with any re-runs in progress at that point discarded.[1]

Summaries of the time-equated results produced by each algorithm on each of the human data sets are shown in Table 2. (`em-indclus` took much longer than the other algorithms and its performance is shown separately in Table 3.) The mean VAF for each algorithm is listed, along with the inter-quartile range (IQR) and the mean number of runs that were necessary to achieve time parity with the slowest algorithm on that data set ($r$). On most instances, there is remarkable variance in the VAF achieved by each algorithm.[2] Overall, despite the variety of approaches that have been brought to bear over the years, the original `indclus` algorithm appears to be the best. (Results in which another algorithm was superior to `indclus` are marked with a box.) Animals-s is the only data set on which its median performance was not the best, and its overall distribution of results is consistently competitive. It is revealing to note the differences in performance between the original and reimplemented versions of `sindclus`. Small changes in the handling of boundary cases make a large difference in the performance of the algorithm.

Surprisingly, on the synthetic data sets (not shown), the relative performance of the algorithms was quite different, and almost the same on the noisy data as on the noise-free data. (This suggests that the randomly generated data sets that are commonly used to evaluate ADCLUS algorithms do not accurately reflect the problems of interest to practitioners.) `ewindclus` performed best here, although it was only occasionally able to recover the original models from the noise-free data.

Overall, it appears that current methods of additive clustering are quite sensitive to the type of problem they are run on and that there is little assurance that they can recover the underlying structure in the data, even for small problems. To address these problems, we turn now to a new approach.

## 3   A Purely Combinatorial Approach

One common theme in `indclus`, `sindclus`, and `ewindclus` is their computation of each cluster and its weight in turn, at each step fitting only the residual similarity not accounted for by the other clusters. This forces memberships to be considered in a predetermined order and allows weights to become obsolete. Inspired in part by recent work of Lee (in press), we propose an orthogonal decomposition of the problem. Instead of computing the elements and weight of each cluster in succession, we first consider all the memberships and then derive all the weights using constrained regression. And where previous algorithms recompute all the memberships of one cluster simultaneously (and therefore independently), we will change memberships one by one in a dynamically determined order using simple heuristic search techniques, recomputing the weights after each step. (An incremental bounded least squares regression algorithm that took advantage of the previous solution would be ideal, but the algorithms tested below did not incorporate such an improvement.) From this perspective, one need only focus on changing the binary membership variables, and ADCLUS becomes a purely combinatorial optimization problem.

We will evaluate three different algorithms based on this approach, all of which attempt to improve a random initial model. The first, `indclus-hc`, is a simple hill-climbing strategy which attempts to toggle individual memberships in an arbitrary order and the first change resulting in an improved model is accepted. The algorithm terminates when no membership can be changed to give an improvement. This strategy is reminiscent of a proposal by Clouse and Cottrell (1996), although here we are using the ADCLUS model of similarity. In the second algorithm, `indclus-pbil`, the PBIL algorithm of Baluja (1997) is used

Table 4: The performance of the combinatorial algorithms on human data sets.

| Name | indclus-hc | | | ind-pbil | | ewind-klb | | | indclus | | |
|---|---|---|---|---|---|---|---|---|---|---|---|
| | VAF | IQR | $\overline{r}$ | VAF | IQR | VAF | IQR | $\overline{r}$ | VAF | IQR | $\overline{r}$ |
| animals-s | 80 | 80–80 | 44 | 74 | 71–74 | 80 | 80–80 | 74 | 80 | 80–80 | 47 |
| numbers | 90 | 90–91 | 24 | 87 | 85–88 | 91 | 91–91 | 18 | 90 | 89–91 | 59 |
| workers | 88 | 88–89 | 16 | 86 | 84–87 | 89 | 89–89 | 13 | 88 | 88–89 | 53 |
| consonants | 86 | 85–87 | 11 | 80 | 76–82 | 92 | 92–92 | 9 | 91 | 91–91 | 61 |
| animals | 71 | 70–72 | 8 | 66 | 65–69 | 74 | 74–74 | 6 | 74 | 74–74 | 36 |
| letters | 70 | 69–71 | 3 | 66 | 64–68 | 76 | 74–78 | 2 | 82 | 81–82 | 57 |

to search for appropriate memberships. This is a simplification of the strategy suggested by Lee (in press), whose proposal also includes elements concerned with automatically controlling model complexity. We use the parameter settings he suggests but only allow the algorithm to generate 10,000 solutions.

### 3.1 KL Break-Out: A New Optimization Heuristic

While the two approaches described above do not use any problem-specific information beyond solution quality, the third algorithm uses the gradient function from the `ewindclus` algorithm to guide the search. The move strategy is a novel combination of gradient ascent and the classic method of Kernighan and Lin (1970) which we call 'KL break-out'. It proceeds by gradient ascent, changing the entry in $F$ whose `ewindclus` gradient points most strongly to the opposite of its current value. When the ascent no longer results in an improvement, a local maximum has been reached. Motivated by results suggesting that good maxima tend to cluster (Boese, Kahng, and Muddu, 1994; Ruml et al., 1996), the algorithm tries to break out of the current basin of attraction and find a nearby maximum rather than start from scratch at another random model. It selects the least damaging variable to change, using the gradient as in the ascent, but now it locks each variable after changing it. The pool of unlocked variables shrinks, thus forcing the algorithm out of the local maximum and into another part of the space. To determine if it has escaped, a new gradient ascent is attempted after each locking step. If the ascent surpasses the previous maximum, the current break-out attempt is abandoned and the ascent is pursued. If the break-out procedure changes all variables without any ascent finding a better maximum, the algorithm terminates. The procedure is guaranteed to return a solution at least as good as that found by the original KL method (although it will take longer), and it has more flexibility to follow the gradient function. This algorithm, which we will call `ewindclus-klb`, surpassed the original KL method in time-equated tests. It is also conceptually simple and has no parameters that need to be tuned.

### 3.2 Evaluation of the Combinatorial Algorithms

The time-equated performance of the combinatorial algorithms on the human data sets is shown in Table 4, with `indclus`, the best of the previous algorithms, shown for comparison. As one might expect, adding heuristic guidance to the search helps it enormously: `ewindclus-klb` surpasses the other combinatorial algorithms on every problem. It performs better than `indclus` on three of the human data sets (top panel), equals its performance on two, and performs worse on one data set, letters. (Results in which `ewindclus-klb` was not the best are marked with a box.) The variance of `indclus` on letters is very small, and the full distributions suggest that `ewindclus-klb` is the better choice on this data set if one can afford the time to take the best of 20 runs. (Experiments

Table 5: `ewindclus-klb` and `indclus` on noisy synthetic data sets of increasing size.

| | ewindclus-klb | | indclus | | |
|---|---|---|---|---|---|
| $n$ | VAF | IQR | VAF | IQR | $\bar{r}$ |
| 8 | 97 | 96–97 | 95 | 93–97 | 1 |
| 16 | 91 | 90–92 | 86 | 85–87 | 4 |
| 32 | 90 | 88–92 | 83 | 82–84 | 22 |
| 64 | 91 | 90–91 | 84 | 84–85 | 100 |
| 128 | 91 | 91–91 | 88 | 87–90 | 381 |

using 7 additional human data sets found that letters represented the weakest performance of `ewindclus-klb`.)

Performance of a plain KL strategy (not shown) surpassed or equaled `indclus` on all but two problems (consonants and letters), indicating that the combinatorial approach, in tandem with heuristic guidance, is powerful even without the new 'KL break-out' strategy.

While we have already seen that synthetic data does not predict the relative performance of algorithms on human data very well, it does provide a test of how well they scale to larger problems. On noise-free synthetic data, `ewindclus-klb` reliably recovered the original model on all data sets. It was also the best performer on the noisy synthetic data (a comparison with `indclus` is presented in Table 5. These results show that, in addition to performing best on the human data, the combinatorial approach retains its effectiveness on larger problems.

In addition to being able to handle larger problems than previous methods, it is important to note that the higher VAF of the models induced by `ewindclus-klb` often translates into increased interpretability. In the model shown in Table 1, for instance, the best previously published model (Tenenbaum, 1996), whose VAF is only 1.6% worse, does not contain š in the unvoiced cluster.

## 4 Conclusions

We formalized the problem of constructing feature-based representations for cognitive modeling as the unsupervised learning of ADCLUS models from similarity data. In an empirical comparison sensitive to variance in solution quality and computation time, we found that several recently proposed methods for recovering such models perform worse than the original `indclus` algorithm of Arabie and Carroll (1980). We suggested a purely combinatorial approach to this problem that is simpler than previous proposals, yet more effective. By changing memberships one at a time, it makes fewer independence assumptions. We also proposed a novel variant of the Kernighan-Lin optimization strategy that is able to follow the gradient function more closely, surpassing the performance of the original.

While this work has extended the reach of the additive clustering paradigm to large problems, it is directly applicable to feature construction of only those cognitive models whose representations encode similarity as shared features. (The cluster weights can be represented by duplicating strong features or by varying connection weights.) However, the simplicity of the combinatorial approach should make it straightforward to extend to models in which the absence of features can enhance similarity. Other future directions include using the output of one algorithm as the starting point for another, and incorporating measures of model complexity(Lee, in press).

# 5 Acknowledgments

Thanks to Josh Tenenbaum, Michael Lee, and the Harvard AI Group for stimulating discussions; to Josh, Anil Chaturvedi, Henk Kiers, J. Douglas Carroll, and Phipps Arabie for providing source code for their algorithms; Josh, Michael, and Phipps for providing data sets; and Michael for sharing unpublished work. This work was supported in part by the NSF under grants CDA-94-01024 and IRI-9618848.

## Footnotes

[1]Depending as it does on running time, this comparison remains imprecise due to variations in the degree of code tuning and the quality of the compilers used, and the need to normalize timings between the multiple machines used in the tests.

[2]Table 3 shows one anomaly: no `em-indclus` run on animals resulted in a VAF $\geq 0$. This also occurred on all synthetic problems with 32 or more objects (although very good solutions were found on the smaller problems). Tenenbaum (personal communication) suggests that the default annealing schedule in the `em-indclus` code may need to be modified for these problems.

# References

Arabie, Phipps and J. Douglas Carroll. 1980. MAPCLUS: A mathematical programming approach to fitting the adclus model. *Psychometrika*, 45(2):211–235, June.

Baluja, Shumeet. 1997. Genetic algorithms and explicit search statistics. In Michael C. Mozer, Michael I. Jordan, and Thomas Petsche, editors, *NIPS 9*.

Boese, Kenneth D., Andrew B. Kahng, and Sudhakar Muddu. 1994. A new adaptive multi-start technique for combinatorial global optimizations. *Operations Research Letters*, 16:101–113.

Carroll, J. Douglas and Phipps Arabie. 1983. INDCLUS: An individual differences generalization of the ADCLUS model and the MAPCLUS algorithm. *Psychometrika*, 48(2):157–169, June.

Chaturvedi, Anil and J. Douglas Carroll. 1994. An alternating combinatorial optimization approach to fitting the INDCLUS and generalized INDCLUS models. *Journal of Classification*, 11:155–170.

Clouse, Daniel S. and Garrison W. Cottrell. 1996. Discrete multi-dimensional scaling. In *Proceedings of the 18th Annual Conference of the Cognitive Science Society*, pp. 290–294.

Hojo, Hiroshi. 1983. A maximum likelihood method for additive clustering and its applications. *Japanese Psychological Research*, 25(4):191–201.

Kernighan, B. and S. Lin. 1970. An efficient heuristic procedure for partitioning graphs. *The Bell System Technical Journal*, 49(2):291–307, February.

Kiers, Henk A. L. 1997. A modification of the SINDCLUS algorithm for fitting the ADCLUS and INDCLUS models. *Journal of Classification*, 14(2):297–310.

Lee, Michael D. in press. A simple method for generating additive clustering models with limited complexity. *Machine Learning*.

Mechelen, I. Van and G. Storms. 1995. Analysis of similarity data and Tversky's contrast model. *Psychologica Belgica*, 35(2–3):85–102.

Noelle, David C., Garrison W. Cottrell, and Fred R. Wilms. 1997. Extreme attraction: On the discrete representation preference of attractor networks. In M. G. Shafto and P. Langley, eds, *Proceedings of the 19th Annual Conference of the Cognitive Science Society*, p. 1000.

Ruml, Wheeler, J. Thomas Ngo, Joe Marks, and Stuart Shieber. 1996. Easily searched encodings for number partitioning. *Journal of Optimization Theory and Applications*, 89(2).

Shepard, Roger N. and Phipps Arabie. 1979. Additive clustering: Representation of similarities as combinations of discrete overlapping properties. *Psychological Review*, 86(2):87–123, March.

Stark, Philip B. and Robert L. Parker. 1995. Bounded-variable least-squares: An algorithm and applications. *Computational Statistics*, 10:129–141.

Tenenbaum, Joshua B. 1996. Learning the structure of similarity. In D. S. Touretzky, M. C. Mozer, and M. E. Hasselmo, editors, *NIPS 8*.
